# Parallel analog VLSI architectures for computation of heading direction and time-to-contact

**Giacomo Indiveri**
giacomo@klab.caltech.edu

**Jörg Kramer**
kramer@klab.caltech.edu

**Christof Koch**
koch@klab.caltech.edu

**Division of Biology**
California Institute of Technology
Pasadena, CA 91125

## Abstract

We describe two parallel analog VLSI architectures that integrate optical flow data obtained from arrays of elementary velocity sensors to estimate heading direction and time-to-contact. For heading direction computation, we performed simulations to evaluate the most important qualitative properties of the optical flow field and determine the best functional operators for the implementation of the architecture. For time-to-contact we exploited the divergence theorem to integrate data from all velocity sensors present in the architecture and average out possible errors.

## 1  Introduction

We have designed analog VLSI velocity sensors invariant to absolute illuminance and stimulus contrast over large ranges that are able to achieve satisfactory performance in a wide variety of cases; yet such sensors, due to the intrinsic nature of analog processing, lack a high degree of precision in their output values. To exploit their properties at a system level, we developed parallel image processing architectures for applications that rely mostly on the qualitative properties of the optical flow, rather than on the precise values of the velocity vectors. Specifically, we designed two parallel architectures that employ arrays of elementary motion sensors for the computation of heading direction and time-to-contact. The application domain that we took into consideration for the implementation of such architectures, is the promising one of vehicle navigation. Having defined the types of images to be analyzed and the types of processing to perform, we were able to use *a priori* infor-

mation to integrate selectively the sparse data obtained from the velocity sensors and determine the qualitative properties of the optical flow field of interest.

## 2   The elementary velocity sensors

A velocity sensing element, that can be integrated into relatively dense arrays to estimate in parallel optical flow fields, has been succesfully built [Kramer *et al.*, 1995]. Unlike most previous implementations of analog VLSI motion sensors, it unambiguously encodes 1-D velocity over considerable velocity, contrast, and illuminance ranges, while being reasonably compact. It implements an algorithm that measures the time of travel of features (here a rapid temporal change in intensity) stimulus between two fixed locations on the chip. In a first stage, rapid dark-to-bright irradiance changes or temporal ON edges are converted into short current pulses. Each current pulse then gives rise to a sharp voltage spike and a logarithmically-decaying voltage signal at each edge detector location. The sharp spike from one location is used to sample the analog voltage of the slowly-decaying signal from an adjacent location. The sampled output voltage encodes the relative time delay of the two signals, and therefore velocity, for the direction of motion where the onset of the slowly-decaying pulse precedes the sampling spike. In the other direction, a lower voltage is sampled. Each direction thus requires a separate output stage.

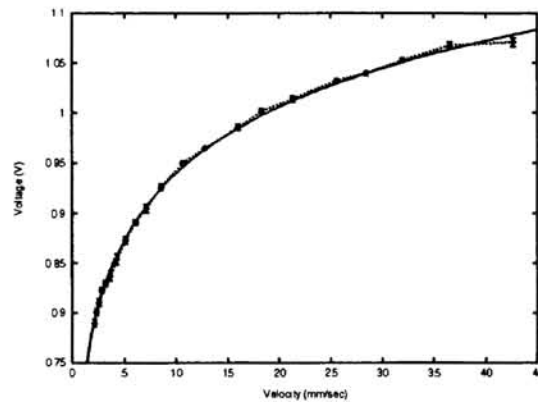

Figure 1: Output voltage of a motion sensing element for the preferred direction of motion of a sharp high-contrast ON edge versus image velocity under incandescent room illumination. Each data point represents the average of 5 successive measurements.

As implemented with a 2 $\mu$m CMOS process, the size of an elementary bi-directional motion element (including 30 transistors and 8 capacitances) is 0.045 mm$^2$. Fig. 1 shows that the experimental data confirms the predicted logarithmic encoding of velocity by the analog output voltage. The data was taken by imaging a moving high-contrast ON edge onto the chip under incandescent room illumination. The calibration of the image velocity in the focal plane is set by the 300 $\mu$m spacing of adjacent photoreceptors on the chip.

## 3   Heading direction computation

To simplify the computational complexity of the problem of heading direction detection we restricted our analysis to pure translational motion, taking advantage of the

fact that for vehicle navigation it is possible to eliminate the rotational component of motion using lateral accelerometer measurements from the vehicle. Furthermore, to analyze the computational properties of the optical flow for typical vehicle navigation scenes, we performed software simulations on sequences of images obtained from a camera with a $64 \times 64$ pixel silicon retina placed on a moving truck (courtesy of B. Mathur at Rockwell Corporation). The optical flow fields have been computed

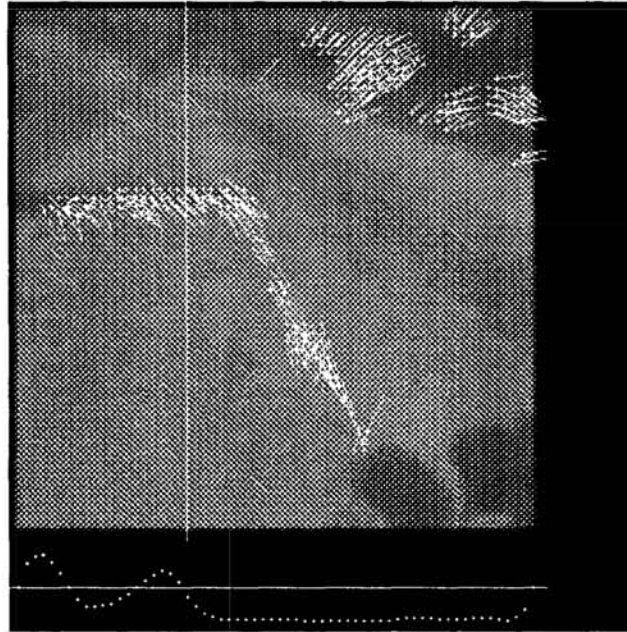

Figure 2: The sum of the horizontal components of the optical flow field is plotted on the bottom of the figure. The presence of more than one zero-crossing is due to different types of noise in the optical flow computation (e.g. quantization errors in software simulations or device mismatch in analog VLSI circuits). The coordinate of the heading direction is computed as the abscissa of the zero-crossing with maximum steepness and closest to the abscissa of the previously selected unit.

by implementing an algorithm based on the *image brightness constancy equation* [Verri *et al.*, 1992] [Barron *et al.*, 1994]. For the application domain considered and the types of optical flow fields obtained from the simulations, it is reasonable to assume that the direction of heading changes smoothly in time. Furthermore, being interested in determining, and possibly controlling, the heading direction mainly along the horizontal axis, we can greatly reduce the complexity of the problem by considering one-dimensional arrays of velocity sensors. In such a case, if we assign positive values to vectors pointing in one direction and negative values to vectors pointing in the opposite direction, the heading direction location will correspond to the point closest to the *zero-crossing*. Under these assumptions, the computation of the horizontal coordinate of the heading direction has been carried out using the following functional operators: thresholding on the horizontal components of the optical flow vectors; spatial smoothing on the resulting values; detection and evaluation of the steepness of the zero-crossings present in the array and finally selection of the zero-crossing with maximum steepness. The zero-crossing with maximum steepness is selected only if its position is in the neighborhood of the previously selected zero-crossing. This helps to eliminate errors due to noise and device mismatch and assures that the computed heading direction location will shift smoothly in time. Fig. 2 shows a result of the software simulations, on an image of a road

with a shadow on the left side.

All of the operators used in the algorithm have been implemented with analog circuits (see Fig. 3 for a block diagram of the architecture). Specifically, we have

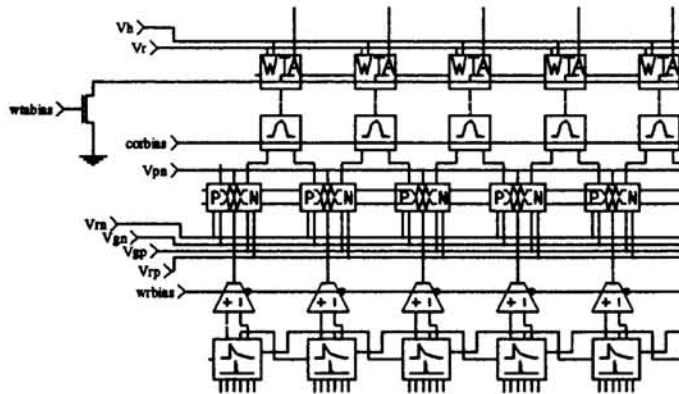

Figure 3: Block diagram of the architecture for detecting heading direction: the first layer of the architecture computes the velocity of the stimulus; the second layer converts the voltage output of the velocity sensors into a positive/negative current performing a threshold operation; the third layer performs a linear smoothing operation on the positive and negative halves of the input current; the fourth layer detects zero-crossings by comparing the intensity of positive currents from one pixel with negative currents from the neighboring pixel; the top layer implements a winner-take-all network with distributed excitation, which selects the zero-crossing with maximum steepness.

designed test chips in which the thresholding function has been implemented using a transconductance amplifier whose current represents the output signal [Mead, 1989], spatial smoothing has been obtained using a circuit that separates positive currents and negative currents into two distinct paths and feeds them into two layers of current-mode diffuser networks [Boahen and Andreou, 1992], the zero-crossing detection and evaluation of its steepness has been implemented using a newly designed circuit block based on a modification of the *simple current-correlator* [Delbrück, 1991], and the selection of the zero-crossing with maximum steepness closest to the previously selected unit has been implemented using a winner-take-all circuit with distributed excitation [Morris *et al.*, 1995]. The schematics of the former three circuits, which implement the top three layers of the diagram of Fig. 3, are shown in Fig. 4.

Fig. 5 shows the output of a test chip in which all blocks up to the diffuser network (without the zero-crossing detection stages) were implemented. The velocity sensor layout was modified to maximize the number of units in the 1-D array. Each velocity sensor measures $60\mu m \times 802\mu m$. On a $(2.2mm)^2$ size chip we were able to fit 23 units. The shown results have been obtained by imaging on the chip expanding or contracting stimuli using black and white edges wrapped around a rotating drum and reflected by an adjacent tilted mirror. The point of contact between drum and mirror corresponding to the simulated heading direction has been imaged approximately onto the $15^{th}$ unit of the array. As shown, the test chip considered does not achieve 100% correct performance due to errors that arise mainly from the presence of parasitic capacitors in the modified part of the velocity sensor circuits; nonetheless, at least from a qualitative point of view, the data confirms the results obtained from software simulations and demonstrates the validity of the approach considered.

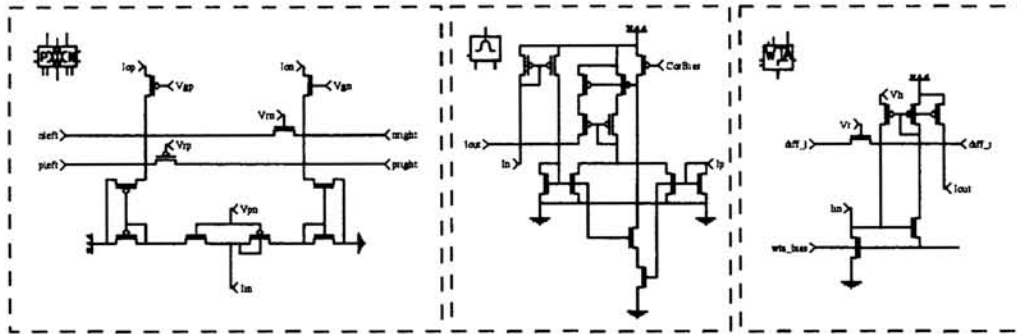

Figure 4: Circuit schematics of the smoothing, zero-detection and winner-take-all blocks respectively.

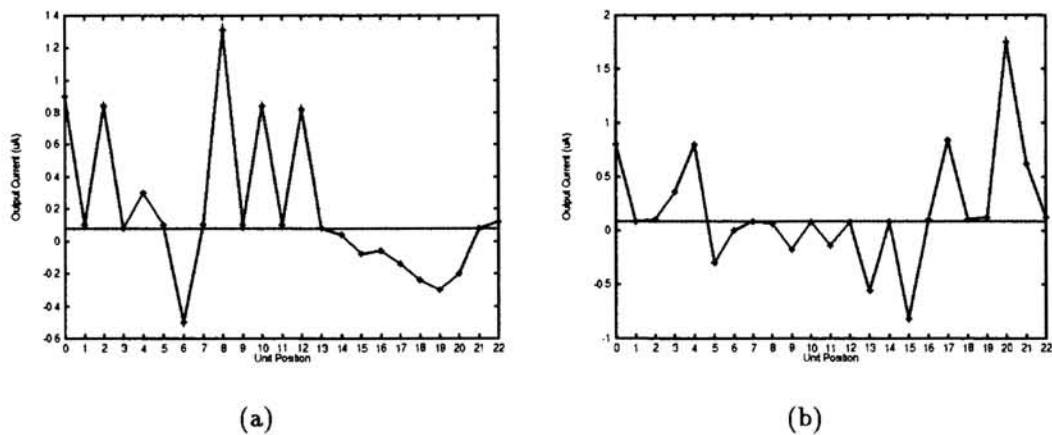

(a)                                                              (b)

Figure 5: Zero crossings computed as difference between smoothed positive currents and smoothed negative currents: (a) for expanding stimuli; (b) for contracting stimuli. The "zero" axis is shifted due to is a systematic offset of 80 nA.

## 4   Time-to-contact

The time-to-contact can be computed by exploiting qualitative properties of the optical flow field such as expansion or contraction [Poggio *et al.*, 1991]. The divergence theorem, or Gauss theorem, as applied to a plane, shows that the integral over a surface patch of the divergence of a vector field is equal to the line integral along the patch boundary of the component of the field normal to the boundary. Since a camera approaching a rigid object sees a linear velocity field, where the velocity vectors are proportional to their distance from the focus-of-expansion, the divergence is constant over the image plane. By integrating the radial component of the optical flow field along the circumference of a circle, the time-to-contact can thus be estimated, independently of the position of the focus-of-expansion.

We implemented this algorithm with an analog integrated circuit, where an array of twelve motion sensing elements is arranged on a circle, such that each element measures velocity radially. According to the Gauss theorem, the time-to-contact is

then approximated by

$$\tau = \frac{N \cdot R}{\sum_{k=1}^{N} v_k} \, , \tag{1}$$

where $N$ denotes the number of elements, $R$ the radius of the circle, and $v_k$ the radial velocity components at the locations of the elements. For each element, temporal aliasing is prevented by comparing the output voltages of the two directions of motion and setting the lower one, corresponding to the null direction, to zero. The output voltages are then used to control subthreshold transistor currents. Since these voltages are logarithmically dependent on velocity, the transistor currents are proportional to the measured velocities. The sum of the velocity components is thus calculated by aggregating the currents from all elements on two lines, one for outward motion and one for inward motion, and taking the difference of the total currents. The resulting bi-directional output current is an inverse function of the signed time-to-contact.

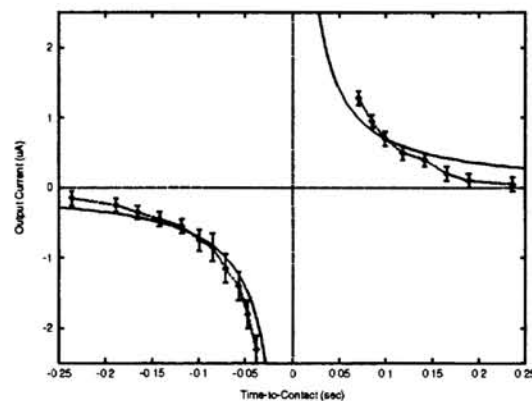

Figure 6: Output current of the time-to-contact sensor as a function of simulated time-to-contact under incandescent room illumination. The theoretical fit predicts an inverse relationship.

The circuit has been implemented on a chip with a size of $(2.2\text{mm})^2$ using 2 $\mu$m technology. The photo diodes of the motion sensing elements are arranged on two concentric circles with radii of 400 $\mu$m and 600 $\mu$m respectively. In order to simulate an approaching or withdrawing object, a high-contrast spiral stimulus was printed onto a rotating disk. Its image was projected onto the chip with a microscope lens under incandescent room illumination. The focus-of-expansion was approximately centered with respect to the photo diode circles. The averaged output current is shown as a function of simulated time-to-contact with a theoretical fit in Fig. 6. The expected inverse relationship is qualitatively observed and the sign (expansion or contraction) is robustly encoded. However, the deviation of the output current from its average can be substantial: Since the output voltage of each motion sensing element decays slowly due to leak currents and since the spiral stimulus causes a serial update of the velocity values along the array, a step change in the output current is observed upon each update, followed by a slow decay. The effect is aggravated, if the individual motion sensing elements measure significantly differing velocities. This is generally the case, because the focus-of-expansion is usually not centered with respect to the sensor and because of inaccuracies in the velocity measurements due to circuit offsets, noise, and the *aperture problem* [Verri *et al.*, 1992]. The integrative property of the algorithm is thus highly desirable, and more robust data would be obtained from an array with more elements and stimuli with higher edge densities.

## 5    Conclusions

We have developed parallel architectures for motion analysis that bypass the problem of low precision in analog VLSI technology by exploiting qualitative properties of the optical flow. The correct functionality of the devices built, at least from a qualitative point of view, have confirmed the validity of the approach followed and induced us to continue this line of research. We are now in the process of designing more accurate circuits that implement the operators used in the architectures proposed.

### Acknowledgments

This work was supported by grants from ONR, ERC and Daimler-Benz AG. The velocity sensor was developed in collaboration with R. Sarpeshkar. The chips were fabricated through the MOSIS VLSI Fabrication Service.

## References

[Barron *et al.*, 1994] J.L. Barron, D.J. Fleet, and S.S. Beauchemin. Performance of optical flow techniques. *International Journal on Computer Vision*, 12(1):43–77, 1994.

[Boahen and Andreou, 1992] K.A. Boahen and A.G. Andreou. A contrast sensitive silicon retina with reciprocal synapses. In *NIPS91 Proceedings*. IEEE, 1992.

[Delbrück, 1991] T. Delbrück. "Bump" circuits for computing similarity and dissimilarity of analog voltages. In *Proc. IJCNN*, pages I–475–479, June 1991.

[Kramer *et al.*, 1995] J. Kramer, R. Sarpeshkar, and C. Koch. An analog VLSI velocity sensor. In *Proc. Int. Symp. Circuit and Systems ISCAS '95*, pages 413–416, Seattle, WA, May 1995.

[Mead, 1989] C.A. Mead. *Analog VLSI and Neural Systems*. Addison-Wesley, Reading, 1989.

[Morris *et al.*, 1995] T.G. Morris, D.M. Wilson, and S.P. DeWeerth. Analog VLSI circuits for manufacturing inspection. In *Conference for Advanced Research in VLSI-Chapel Hill, North Carolina*, March 1995.

[Poggio *et al.*, 1991] T. Poggio, A. Verri, and V. Torre. Green theorems and qualitative properties of the optical flow. Technical report, MIT, 1991. Internal Lab. Memo 1289.

[Verri *et al.*, 1992] A. Verri, M. Straforini, and V. Torre. Computational aspects of motion perception in natural and artificial vision systems. *Phil. Trans. R. Soc. Lond. B*, 337:429–443, 1992.
